# Connectionist Implementation of a Theory of Generalization

**Roger N. Shepard**
Department of Psychology
Stanford University
Stanford, CA 94305-2130

**Sheila Kannappan**
Department of Physics
Harvard University
Cambridge, MA 02138

## Abstract

Empirically, generalization between a training and a test stimulus falls off in close approximation to an exponential decay function of distance between the two stimuli in the "stimulus space" obtained by multidimensional scaling. Mathematically, this result is derivable from the assumption that an individual takes the training stimulus to belong to a "consequential" region that includes that stimulus but is otherwise of unknown location, size, and shape in the stimulus space (Shepard, 1987). As the individual gains additional information about the consequential region—by finding other stimuli to be consequential or not—the theory predicts the shape of the generalization function to change toward the function relating actual probability of the consequence to location in the stimulus space. This paper describes a natural connectionist implementation of the theory, and illustrates how implications of the theory for generalization, discrimination, and classification learning can be explored by connectionist simulation.

## 1  THE THEORY OF GENERALIZATION

Because we never confront exactly the same situation twice, anything we have learned in any previous situation can guide us in deciding which action to take in the present situation only to the extent that the similarity between the two situations is sufficient to justify generalization of our previous learning to the present situation. Accordingly, principles of generalization must be foundational for any theory of behavior.

In Shepard (1987) nonarbitrary principles of generalization were sought that would be optimum in any world in which an object, however distinct from other objects, is generally a member of some class or *natural kind* sharing some dispositional property of potential consequence for the individual. A newly encountered plant or animal might be edible or

poisonous, for example, depending on the hidden genetic makeup of its natural kind.

This simple idea was shown to yield a quantitative explanation of two very general empirical regularities that emerge when generalization date are submitted to methods of *multidimensional scaling*. The first concerns the shape of the *generalization gradient*, which describes how response probability falls off with distance of a test stimulus from the training stimulus in the obtained representational space. The second, which is not treated in the present (unidimensional) connectionist implementation, concerns the metric of multidimensional representational spaces. (See Shepard, 1987.)

These results were mathematically derived for the simplest case of an individual who, in the absence of any advance knowledge about particular objects, now encounters one such object and discovers it to have an important consequence. From such a learning event, the individual can conclude that all objects are consequential that are of the same kind as that object and that therefore fall in some *consequential region* that overlaps the point corresponding to that object in representational space. The individual can only estimate the probability that a given new object is consequential as the conditional probability, given that a region of unknown size and shape overlaps that point, that it also overlaps the point corresponding to the new object. The gradient of generalization then arises because a new object that is closer to the old object in the representational space is more likely to fall within a random region that overlaps the old object.

In order to obtain a quantitative estimate of the probability that the new stimulus is consequential, the individual must integrate over all candidate regions in representational space—with, perhaps, different probabilities assigned, a priori, to different sizes and shapes of region. The results turn out to depend remarkably little on the prior probabilities assigned (Shepard, 1987). For any reasonable choice of these probabilities, integration yields an approximately exponential gradient. And, for the single most reasonable choice in the absence of any advance information about size or shape, namely, the choice of maximum entropy prior probabilities, integration yields exactly the exponential decay function.

These results were obtained by separating the *psychological* problem of the form of generalization in a psychological space from the *psychophysical* problem of the mapping from any physical parameter space to that psychological space. The psychophysical mapping, having been shaped by natural selection, would favor a representational space in which regions that correspond to natural kinds, though variously sized and shaped, are not on average systematically elogated or compressed in any particular direction or location of the space. Such a regularized space would provide the best basis for generalization from objects of newly encountered kinds.

The psychophysical mapping thus corresponds to an optimum mapping from input to hidden units in a connectionist system. Indeed, Rumelhart (1990) has recently suggested that the power of the connectionist approach comes from the ability of a set of hidden units to represent the relations among possible inputs according to their significances for the system as a whole rather than according to their superficial relations at the input level. Although in biologically evolved individuals the psychophysical mapping is likely to have been shaped more through evolution than through learning (Shepard, 1989; see also Miller & Todd, 1990) the connectionist implementation to be described here does provide for some fine tuning of this mapping through learning.

Beyond the exponential form of the gradient of generalization following training on a single stimulus, three basic phenomena of discrimination and classification learning that

the theory of generalization should be able to explain are the following: First, when all and only the stimuli within a compact subset are followed by an important consequence (reinforcement), an individual should eventually learn to respond to all and only the stimuli in that subset (Shepard, 1990)—at least to the degree possible, given any noise-induced uncertainty about locations in the representational space (Shepard, 1986, 1990). Second, when the positive stimuli do not form a compact subset but are interspersed among negative (nonreinforced) stimuli, generalization should entail a slowing of classification learning (Nosofsky, 1986; Shepard & Chang, 1963). Third, repeated discrimination or classification learning, in which a boundary between positive and negative stimuli remains fixed, should induce a "fine tuning" stretching of the representational space at that boundary such that any subsequent learning will generalize less fully across that boundary.

Our initial connectionist explorations have been for relatively simple cases using a unidemensional stimulus set and a linear learning rule. These simulations serve to illustrate how information about the probable disposition of a consequential region accrues, in a Bayesian manner, from successive encounters with different stimuli, each of which is or is not followed by the consequence. In complex cases, the cumulative effects on probability of generalized response, on latency of discriminative response, and on fine tuning of the psychophysical mapping may sometimes be easier to establish by simulation than by mathematical derivation. Fortunately for this purpose, the theory of generalization has a connectionist embodiment that is quite direct and neurophysiologically plausible.

## 2    A CONNECTIONIST IMPLEMENTATION

In the implementation reported here, a linear array of 20 input units represents a set of 20 stimuli differing along a unidimensional continnuum, such as the continuum of pitches of tones. The activation level of a given input unit is 1 when its corresponding stimulus is presented and 0 when it is not. (This localist representation of the "input" may be considered the output of a lower-level, massively parallel network for perpetual analysis.)

When such an "input unit" is activated, its activation propagates upward and outward through successively higher layers of hidden units, giving rise to a *cone of activation* of that input unit (Figure 1a). Higher units are activated by wider ranges of input units (i.e., have larger "receptive fields"). The hidden units thus represent potential consequential regions, with higher units corresponding to regions of greater sizes in representational space.

The activation from any input unit is also subject to progressive attenuation as it propagates to succesively higher layers of hidden units. In the present form of the model, this attenuation comes about because the weights of the connections from input to hidden units fall off exponentially with the heights of the hidden units. (Connection weights are pictorially indicated in Figure 1 by the heavinesses of the connecting lines.) An exponential fall off of connection weight with height is natural, in that it corresponds to a decrement of fixed proportion as the activation propagates through each layer to the next. According to the generalizaton theory (Shepard, 1987), an exponential fall off is also optimum for the case of minimum prior knowledge, because it corresponds to the maximum entropy probability density distribution of possible sizes of a consequential region.

When a response, $R_k$, is followed by a positive consequence in the presence of a stimulus, $S_1$, a simple linear rule (either a Hebbian or a delta rule) will increase the weight of the connection from each representational unit, $j$, (whether input or hidden unit) to that response

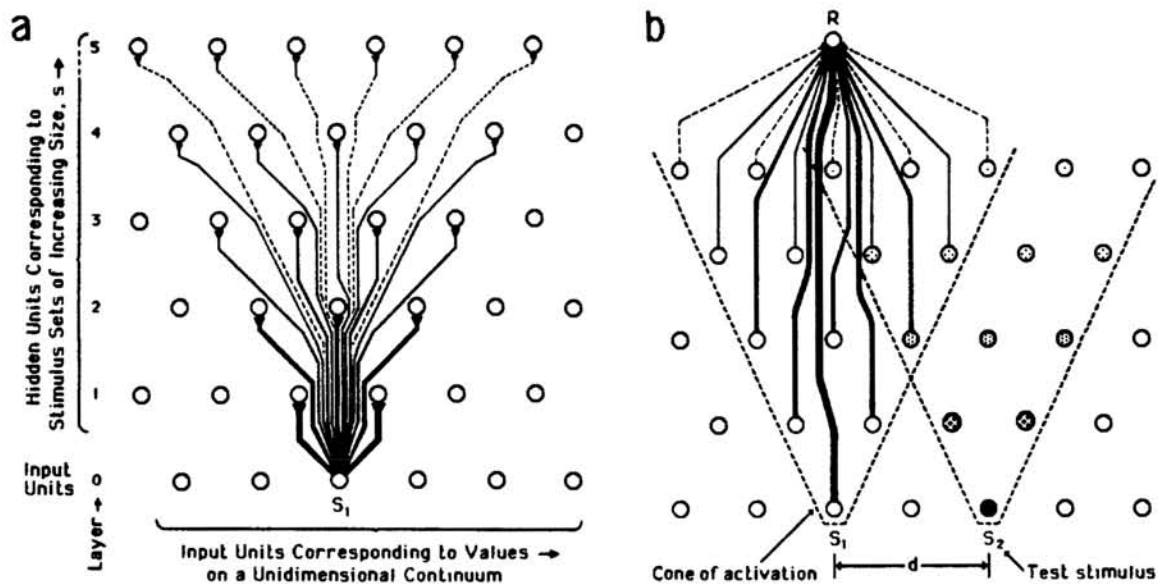

Figure 1: Schematic portrayal of the connectionist embodiment. (a) Initial connections from an input unit to hidden units in its "cone of activation." (b) Connections from these hidden units to a response unit following reinforcement of the response.

unit, $R_k$, in proportion to the current level of activation, $a_j$, of that representational unit. In the initial implementation considered here, the change in weight from representational unit $j$ to the response unit $R_k$ is simply

$$\Delta w_{jk} = \begin{cases} \lambda a_j (1 - a_k) & \text{upon a positive outcome (reinforcement)} \\ -\lambda a_j a_k & \text{upon a negative outcome (nonreinforcement)} \end{cases}$$

where $\lambda$ is a learning rate parameter and $a_k$ is the current activation level of the response unit $R_k$ (which, tending to be confined between 0 and 1, represents an estimate of the probability of the positive consequence). Following a positive outcome, then, positive weights will connect all the units in the cone of activation for $S_1$ to $R_k$, but with values that decay exponentially with the height of a unit in that cone (Figure 1b).

If, now, a different stimulus, $S_2$, is encountered, some but not all of the representational units that are in the cone of activation of $S_1$ and, hence, that are already connected to $R_k$ will also fall in the cone of activation of $S_2$ (Figure 1b). It is these units in the overlap of the two cones that mediate generalization of the response from $S_1$ to $S_2$. Not only is this simple connectionist scheme neurophysiologically plausible, it is also isomorphic to the theory of generalization (Shepard, 1987) based solely on considerations of optimal behavior in a world consisting of natural kinds.

## 3   PRELIMINARY CONNECTIONIST EXPLORATIONS

The simulation results for generalization and discrimination learning are summarized in Figure 2. Panel a shows, for different stages of training on stimulus $S_{10}$, the level of response activation produced by activation of each of the 20 input units. In accordance with theory, this activation decayed exponentially with distance from the training stimulus. The obtained functions differ only by a multiplicative scale factor that increased (toward asymptote) with the amount of training. Following this training, the response connection weights decreased exponentially with the heights of the hidden units (Panel b). Later training on a second positive stimulus, $S_{12}$, created a secondary peak in the activation function (Panel c), and still later nonreinforced presentation of a third stimulus, $S_9$, produced a sharp drop in the activation function at the discrimination boundary (Panel d).

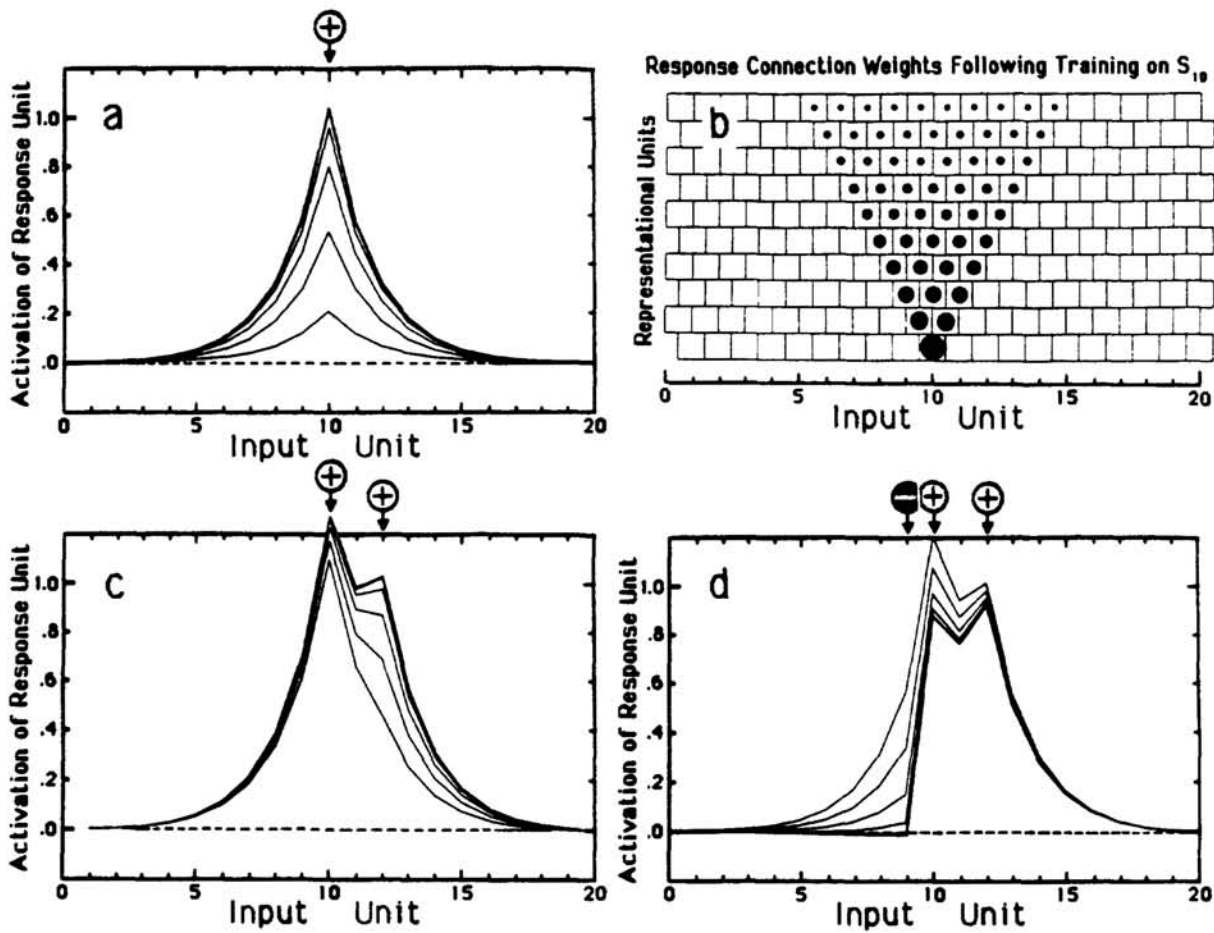

Figure 2: Connectionist simulations of generalization and discrimination learning.

Figure 3 presents the results for classification learning in which all stimuli were presented but with response reinforcement for stimuli in the positive set only. When the positive set was compact (Panel a) sharp discrimination boundaries formed and response activation approached 1 for all positive stimuli and 0 for all negative stimuli. In accordance with theory and empirical data, generalization entailed slower classification learning when the positive stimuli were dispersed among negative stimuli (Panel b)—as shown by a (mean square) error measure (Panel c).

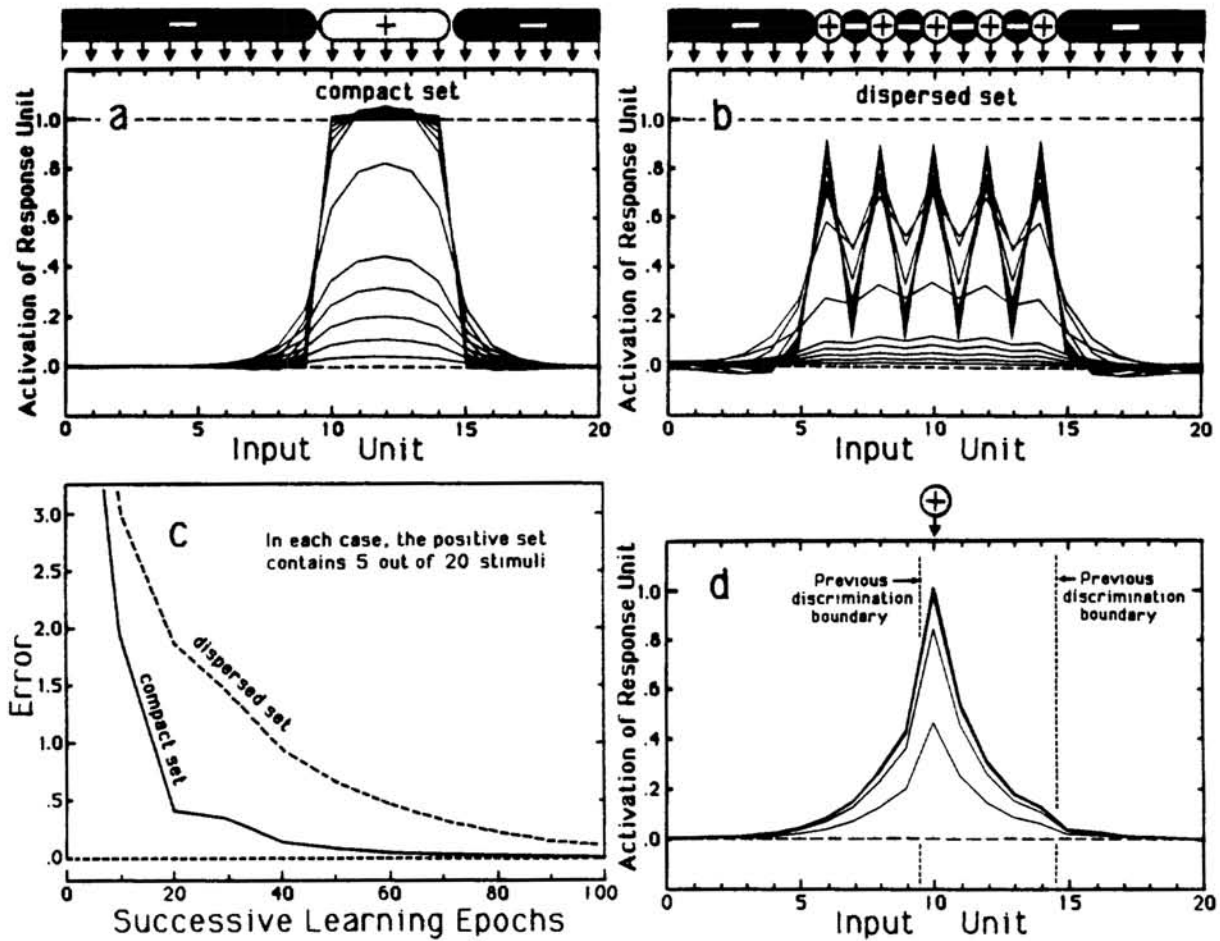

Figure 3: Connectionist simulations of classification learning.

Finally, Panel d illustrates fine tuning of the psychophysical mapping when discrimination boundaries have the same locations for many successively learned classifications. In contrast to the preceding simulations, in which only the response connection weights were allowed to change, here the connection weights from the input units to the hidden units were also allowed to change through "back propagation" (Rumelhart, Hinton, & Williams, 1986). For 400 learning epochs each, each of ten different responses was successively associated with the same five positive stimuli, $S_{10}$ through $S_{14}$, while reinforcement was withheld for all the remaining stimuli. Then, yet another response was associated with the single stimulus $S_{10}$. Although the resulting activation curves for this new response (Panel d) are similar to the original generalization curves (Figure 2a), they drop more sharply where classification boundaries were previously located. This fine tuning of the psychophysical mapping proceeded, however, much more slowly than the learning of the classificatory responses themselves.

## 4   CONCLUDING REMARKS

This is just the beginning of the connectionist exploration of the implications of the generalization theory in more complex cases. In addition to accounting for generalization

and classification along a unidimensional continuum, the approach can account for generalization and classification of stimuli differing with respect to multidimensional continua (Shepard, 1987) and also with respect to discrete features (Gluck, 1991; Russell, 1986). Finally, the connectionist implementation should facilitate a proposed extension to the treatment of response latencies as well as probabilities (Shepard, 1987).

Connectionists have sometimes assumed an exponential decay generalization function, and their notion of radial basis functions is not unlike the present concept of consequential regions (see Hanson & Gluck, this volume). What has been advocated here (and in Shepard, 1987) is the derivation of such functions and concepts from first principles.

## Acknowledgements

This work was supported by National Science Foundation grant BNS85-11685 to the first author. For help and guidance, we thank Jonathan Bachrach, Geoffrey Miller, Mark Monheit, David Rumelhart, and Steven Sloman.

## References

Gluck, M. A. (1991). Stimulus generalization and representation in adaptive network models of category learning. *Psychological Science, 2*. (in press).

Hanson, S. J. & Gluck, M. A. (1991). Spherical units as dynamic consequential regions: Implications for attention, competition, and categorization. (This volume).

Miller, G. F. & Todd, P. M. (1990). Exploring adaptive agency I: Theory and methods for simulating the evolution of learning. In D. S. Touretzky, J. L. Elman, T. J. Sejnowski, & G. E. Hinton (Eds.), *Proceedings of the 1990 Connectionist Models Summer School*. San Mateo, CA: Morgan Kaufmann.

Nosofsky, R. M. (1986). Attention, similarity, and the identification-categorization relationship. *Journal of Experimental Psychology: General, 114*, 39–57.

Rumelhart, D. E. (1990). Representation in connectionist models (The Association Lecture). *Attention & Performance Meeting*. Ann Arbor, Michigan, July 9.

Rumelhart, D. E., Hinton, G. E., & Williams, R. J. (1986). Learning representations by back-propagating errors. *Nature, 323*, 533–536.

Russell, S. J. (1986). A quantitative analysis of analogy by similarity. In *Proceedings of the National Conference on Artificial Intelligence*. Philadelphia, PA: American Association for Artificial Intelligence.

Shepard, R. N. (1986). Discrimination and generalization in identification and classification: Comment on Nosofsky. *Journal of Experimental Psychology: General, 115*, 50–61.

Shepard, R. N. (1987). Toward a universal law of generalization for psychological science. *Science, 237*, 1317–1323.

Shepard, R. N. (1989). Internal representation of universal regularities: A challenge for connectionism. In L. Nadel, L. A. Cooper, P. Culicover, & R. M. Harnish (Eds.), *Neural Connections, Mental Computation* (pp. 103–104). Cambridge, MA: MIT Press.

Shepard, R. N. (1990). Neural nets for generalization and classification: Comment on Staddon and Reid. *Psychological Review, 97*, 579–580.

Shepard, R. N. & Chang, J. J. (1963). Stimulus generalization in the learning of classifications. *Journal of Experimental Psychology, 65*, 94–102.


